# Near-Minimax Optimal Classification with Dyadic Classification Trees

**Clayton Scott**
Electrical and Computer Engineering
Rice University
Houston, TX 77005
cscott@rice.edu

**Robert Nowak**
Electrical and Computer Engineering
University of Wisconsin
Madison, WI 53706
nowak@engr.wisc.edu

## Abstract

This paper reports on a family of computationally practical classifiers that converge to the Bayes error at near-minimax optimal rates for a variety of distributions. The classifiers are based on *dyadic classification trees* (DCTs), which involve adaptively pruned partitions of the feature space. A key aspect of DCTs is their spatial adaptivity, which enables local (rather than global) fitting of the decision boundary. Our risk analysis involves a spatial decomposition of the usual concentration inequalities, leading to a spatially adaptive, data-dependent pruning criterion. For any distribution on $(X, Y)$ whose Bayes decision boundary behaves locally like a Lipschitz smooth function, we show that the DCT error converges to the Bayes error at a rate within a logarithmic factor of the minimax optimal rate. We also study DCTs equipped with polynomial classification rules at each leaf, and show that as the smoothness of the boundary increases their errors converge to the Bayes error at a rate approaching $n^{-1/2}$, the parametric rate. We are not aware of any other practical classifiers that provide similar rate of convergence guarantees. Fast algorithms for tree pruning are discussed.

## 1 Introduction

We previously studied *dyadic classification trees*, equipped with simple binary decision rules at each leaf, in [1]. There we applied standard structural risk minimization to derive a pruning rule that minimizes the empirical error plus a complexity penalty proportional to the square root of the size of the tree. Our main result concerned the rate of convergence of the expected error probability of our pruned dyadic classification tree to the Bayes error for a certain class of problems. This class, which essentially requires the Bayes decision boundary to be locally Lipschitz, had previously been studied by Mammen and Tsybakov [2]. They showed the minimax rate of convergence for this class to be $n^{-1/d}$, where $n$ is the number of labeled training samples, and $d$ is the dimension of each sample. They also demonstrated a classification rule achieving this rate, but the rule requires minimization of the empirical error over the entire class of decision boundaries, an infeasible task in practice. In contrast, DCTs are computationally efficient, but converge at a slower rate of $n^{-1/(d+1)}$.

In this paper we exhibit a new pruning strategy that is both computationally efficient and realizes the minimax rate to within a log factor. Our approach is motivated by recent results from Kearns and Mansour [3] and Mansour and McAllester [4]. Those works develop a theory of local uniform convergence, which allows the error to be decomposed in a spatially adaptive way (unlike conventional structural risk minimization). In essence, the associated pruning rules allow a more refined partition in a region where the classification problem is harder (i.e., near the decision boundary). Heuristic arguments and anecdotal evidence in both [3] and [4] suggest that spatially adaptive penalties lead to improved performance compared to "global" penalties. In this work, we give theoretical support to this claim (for a specific kind of classification tree, the DCT) by showing a superior rate of convergence for DCTs pruned according to spatially adaptive penalties.

We go on to study DCTs equipped with polynomial classification rules at each leaf. This provides more flexible classifiers that can take advantage of additional smoothness in the Bayes decision boundary. We call such a classifier a polynomial-decorated DCT (PDCT). PDCTs can be practically implemented by employing polynomial kernel SVMs at each leaf node of a pruned DCT. For any distribution whose Bayes decision boundary behaves locally like a Hölder-$\gamma$ smooth function, we show that the PDCT error converges to the Bayes error at a rate no slower than $O((\log n/n)^{\gamma/(d+2\gamma-2)})$. As $\gamma \to \infty$ the rate tends to within a log factor of the parametric rate, $n^{-1/2}$.

*Perceptron trees*, tree classifiers having linear splits at each node, have been investigated by many authors and in particular we point to the works [5,6]. Those works consider optimization methods and generalization errors associated with perceptron trees, but do not address rates of approximation and convergence. A key aspect of PDCTs is their spatial adaptivity, which enables local (rather than global) polynomial fitting of the decision boundary. Traditional polynomial kernel-based methods are not capable of achieving such rates of convergence due to their lack of spatial adaptivity, and it is unlikely that other kernels can solve this problem for the same reason. Consider approximating a Hölder-$\gamma$ smooth function on a bounded domain with a single polynomial. Then the error in approximation is $O(1)$, a constant, which is the best one could hope for in learning a Hölder smooth boundary with a traditional polynomial kernel scheme. On the other hand, if we partition the domain into hypercubes of side length $O(1/m)$ and fit an individual polynomial on each hypercube, then the approximation error decays like $O(m^{-\gamma})$. Letting $m$ grow with the sample size $n$ guarantees that the approximation error will tend to zero. On the other hand, pruning back the partition helps to avoid overfitting. This is precisely the idea behind the PDCT.

## 2   Dyadic Classification Trees

In this section we review our earlier results on dyadic classification trees. Let $X$ be a $d$-dimensional observation, and $Y \in \{0, 1\}$ its class label. Assume $X \in [0, 1]^d$. This is a realistic assumption for real-world data, provided appropriate translation and scaling has been applied. DCTs are based on the concept of a *cyclic dyadic partition* (CDP). Let $\mathcal{P} = \{R_1, \ldots, R_k\}$ be a tree-structured partition of the input space, where each $R_i$ is a hyperrectangle with sides parallel to the coordinate axes. Given an integer $\ell$, let $[\ell]_d$ denote the element of $\{1, \ldots, d\}$ that is congruent to $\ell$ modulo $d$. If $R_i \in \mathcal{P}$ is a cell at depth $j$ in the tree, let $R_i^{(1)}$ and $R_i^{(2)}$ be the rectangles formed by splitting $R_i$ at its midpoint along coordinate $[j + 1]_d$. A CDP is a partition $\mathcal{P}$ constructed according to the rules: (i) The trivial partition $\mathcal{P} = [0, 1]^d$ is a CDP; (ii) If $\{R_1, \ldots, R_k\}$ is a CDP, then so is $\{R_1, \ldots, R_{i-1}, R_i^{(1)}, R_i^{(2)}, R_{i+1}, \ldots, R_k\}$, where $1 \leq i \leq d$. The term "cyclic" refers to how the splits cycle through the coordinates of the input space as one traverses a path down the tree. We define a *dyadic classification tree* (DCT) to be a cyclic dyadic partition with

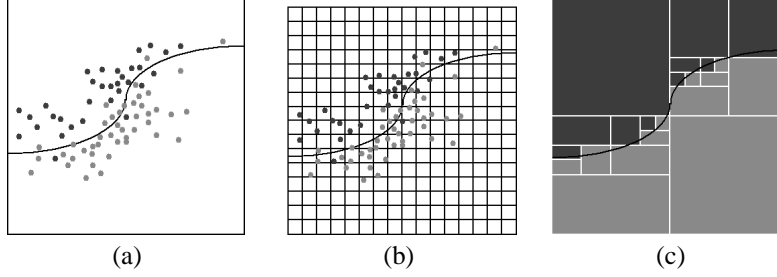

| (a) | (b) | (c) |

Figure 1: Example of a dyadic classification tree when $d = 2$. (a) Training samples from two classes, and Bayes decision boundary. (b) Initial dyadic partition. (c) Pruned dyadic classification tree. Polynomial-decorated DCTs, discussed in Section 4, are similar in structure, but a polynomial decision rule is employed at each leaf of the pruned tree, instead of a simple binary label.

a class label (0 or 1) assigned to each node in the tree. We use the notation $T$ to denote a DCT. Figure 1 (c) shows an example of a DCT in the two-dimensional case.

Previously we presented a rule for pruning DCTs with consistency and rate of convergence properties. In this section we review those results, setting the stage for our main result in the next section. Let $m = 2^J$ be a dyadic integer, and define $T_0$ to be the DCT that has every leaf node at depth $dJ$. Then each leaf of $T_0$ corresponds to a cube of side length $1/m$, and $T_0$ has $m^d$ total leaf nodes. Assume a training sample of size $n$ is given, and each node of $T_0$ is labeled according to a majority vote with respect to the training data reaching that node. A subtree $T$ of $T_0$ is referred to as a *pruned* subtree, denoted $T \leq T_0$, if $T$ includes the root of $T_0$, if every internal node of $T$ has both its children in $T$, and if the nodes of $T$ inherit their labels from $T_0$. The *size* of a tree $T$, denoted $|T|$, is the number of leaf nodes.

We defined the complexity penalized dyadic classification tree $T'_n$ to be the solution of

$$T'_n = \operatorname*{arg\,min}_{T \leq T_0} \ \hat{\epsilon}(T) + \alpha_n \sqrt{|T|}, \tag{1}$$

where $\alpha_n = \sqrt{32 \log(en)/n}$, and $\hat{\epsilon}(T)$ is the empirical error, i.e., the fraction of training data misclassified by $T$. (The solution to this pruning problem can be computed efficiently [7].) We showed that if $X \in [0,1]^d$ with probability one, and $m^d = o(n/\log n)$, then $\mathbf{E}\{\epsilon(T'_n)\} \to \epsilon^*$ with probability one (i.e., $T'_n$ is consistent). Here, $\epsilon(T) = \mathbf{P}\{T(X) \neq Y\}$ is the true error probability for $T$, and $\epsilon^*$ is the Bayes error, i.e., the minimum error probability over all classifiers (not just trees). We also demonstrated a rate of convergence result for $T'_n$, under certain assumptions on the distribution of $(X, Y)$. Let us recall the definition of this class of distributions. Again, let $X \in [0,1]^d$ with probability one.

**Definition 1** *Let $c_1, c_2 > 0$, and let $m_0$ be a dyadic integer. Define $\mathcal{F} = \mathcal{F}(c_1, c_2, m_0)$ to be the collection of all distributions on $(X, Y)$ such that*

**A1** (Bounded density): *For any measurable set $A$, $\mathbf{P}\{X \in A\} \leq c_1 \lambda(A)$, where $\lambda$ denotes the Lebesgue measure.*

**A2** (Regularity): *For all dyadic integers $m \geq m_0$, if we subdivide the unit cube into cubes of side length $1/m$, The Bayes decision boundary passes through at most $c_2 m^{d-1}$ of the resulting $m^d$ cubes.*

These assumptions are satisfied when the density of $X$ is essentially bounded with respect to Lebesgue measure, and when the Bayes decision boundary for the distribution on $(X, Y)$ behaves locally like a Lipschitz function. See, for example, the boundary fragment class of [2] with $\gamma = 1$ therein.

In [1], we showed that if the distribution of $(X, Y)$ belongs to $\mathcal{F}$, and $m \sim (n/\log n)^{1/(d+1)}$, then $\mathbf{E}\{\epsilon(T'_n)\} - \epsilon^* = O((\log n/n)^{1/(d+1)})$. However, this upper bound on the rate of convergence is not tight. The results of Mammen and Tsybakov [2] show that the minimax rate of convergence, $\inf_{\phi_n} \sup_{\mathcal{F}} \mathbf{E}\{\epsilon(\phi_n)\} - \epsilon^*$, is on the order of $n^{-1/d}$ (here $\phi_n$ ranges over all possible discrimination rules). In the next section, we introduce a new strategy for pruning DCTs, which leads to an improved rate of convergence of $(\log n/n)^{1/d}$ (i.e., within a logarithmic factor of the minimax rate). We are not aware of other practically implementable classifiers that can achieve this rate.

## 3 Improved Tree Pruning with Spatially Adaptive Penalties

An improved rate of convergence is achieved by pruning the initial tree $T_0$ using a new complexity penalty. Given a node $v$ in a tree $T$, let $T_v$ denote the subtree of $T$ rooted at $v$. Let $S$ denote the training data, and let $n_v$ denote the number of training samples reaching node $v$. Let $R$ denote a pruned subtree of $T$. In the language of [4], $R$ is called a *root fragment*. Let $L(R)$ denote the set of leaf nodes of $R$.

Consider the pruning rule that selects

$$T_n = \underset{T \leq T_0}{\arg\min} \left( \hat{\epsilon}(T) + \min_{R \leq T} \Delta(T, S, R) \right), \tag{2}$$

where

$$\Delta(T, S, R) = \sum_{v \in L(R)} \frac{1}{n} \left[ \sqrt{48 n_v |T_v| \log(2n)} + \sqrt{48 n_v d \log(m)} \right].$$

Observe that the penalty is data-dependent (since $n_v$ depends on $S$) and spatially adaptive (choosing $R \leq T$ to minimize $\Delta$). The penalty can be interpreted as follows. The first term in the penalty is written $\sum_{v \in L(R)} \hat{p}_v \sqrt{48 |T_v| \log(2n)/n_v}$, where $\hat{p}_v = n_v/n$. This can be viewed as an empirical average of the complexity penalties for each of the subtrees $T_v$, which depend on the *local* data associated with each subtree. The second term can be interpreted as the "cost" of spatially decomposing the bound on the generalization error.

The penalty has the following property. Consider pruning one of two subtrees, both with the same size, and assume that both options result in the same increase in the empirical error. Then the subtree with more data is selected for pruning. Since deeper nodes typically have less data, this shows that the penalty favors unbalanced trees, which may promote higher resolution (deeper leaf nodes) in the vicinity of the decision boundary. In contrast, the pruning rule (1) penalizes balanced and unbalanced trees (with the same size) equally.

The following theorem bounds the expected error of $T_n$. This kind of bound is known as an index of resolvability result [3,8]. Recall that $m$ specifies the depth of the initial tree $T_0$.

**Theorem 1** *If $m \sim (n/\log n)^{1/d}$, then*

$$\mathbf{E}\{\epsilon(T_n) - \epsilon^*\} \leq \min_{T \leq T_0} \left\{ (\epsilon(T) - \epsilon^*) + \mathbf{E}\left[\min_{R \leq T} \Delta(T, S, R)\right] \right\} + O\left( \sqrt{\frac{\log n}{n}} \right).$$

The first term in braces on the right is the approximation error. The remaining terms on the right-hand side bound the estimation error. Since the bound holds for all $T$, one feature of the pruning rule (2) is that $T_n$ performs at least as well as the subtree $T \leq T_0$ that minimizes the bound. This theorem may be applied to give us our desired rate of convergence result.

**Theorem 2** *Assume the density of $(X, Y)$ belongs to $\mathcal{F}$. If $m \sim (n/\log n)^{1/d}$, then*

$$\mathbf{E}\{\epsilon(T_n)\} - \epsilon^* = O((\log n/n)^{1/d}).$$

In other words, the pruning rule (2) comes within a log factor of the minimax rate. These theorems are proved in the last section.

## 4 Faster Rates for Smoother Boundaries

In this section we extend Theorem 2 to the case of smoother decision boundaries. Define $\mathcal{F}(\gamma, c_1, c_2, m_0) \subset \mathcal{F}(c_1, c_2, m_0)$ to be those distributions on $(X, Y)$ satisfying the following additional assumption. Here $\gamma \geq 1$ is fixed.

**A3** ($\gamma$-regularity): *Subdivide $[0,1]^d$ into cubes of side length $1/m$, $m \geq m_0$. Within each cube the Bayes decision boundary is described by a function (one coordinate is a function of the others) with Hölder regularity $\gamma$.*

The collection $\mathcal{G}$ contains all distributions whose Bayes decision boundaries behave locally like the graph of a function with Hölder regularity $\gamma$. The "boundary fragments" class of Mammen and Tsybakov is a special case of boundaries satisfying **A1** and **A3**.

We propose a classifier, called a polynomial-decorated dyadic classification tree (PDCT), that achieves fast rates of convergence for distributions satisfying A3. Given a positive integer $r$, a PDCT of degree $r$ is a DCT, with class labels at each leaf node assigned by a degree $r$ polynomial classifier.

Consider the pruning rule that selects

$$T_{n,r} = \underset{T \leq T_0}{\arg \min} \left( \hat{\epsilon}(T) + \min_{R \leq T} \Delta^r(T, S, R) \right), \tag{3}$$

where

$$\Delta^r(T, S, R) = \sum_v \frac{1}{n} \left[ \sqrt{48 n_v V_{d,r} |T_v| \log(2n)} + \sqrt{48 n_v (d + \gamma) \log(m)} \right].$$

Here $V_{d,r} = \binom{d+r}{r}$ is the $VC$ dimension of the collection of degree $r$ polynomial classifiers in $d$ dimensions. Also, the notation $T \leq T_0$ in (3) is rough. We actually consider a search over all pruned subtrees of $T_0$, and with all possible configurations of degree $r$ polynomial classifiers at the leaf nodes.

An index of resolvability result analgous to Theorem 1 for $T_{n,r}$ can be derived. Moreover, If $r = \lceil \gamma \rceil - 1$, then a decision boundary with Hölder regularity $\gamma$ is well approximated by a PDCT of degree $r$. In this case, $T_{n,r}$ converges to the Bayes risk at rates bounded by the next theorem.

**Theorem 3** *Assume the density of $(X, Y)$ belongs to $\mathcal{G}$ and that $r = \lceil \gamma \rceil - 1$. If $m \sim (n/\log n)^{1/(d+2\gamma-2)}$, then*

$$\mathbf{E}\{\epsilon(T_{n,r})\} - \epsilon^* = O((\log n/n)^{\gamma/(d+2\gamma-2)}).$$

Note that in the case $\gamma = 1$ this result coincides with the near-minimax rate in Theorem 2. Also notice that as $\gamma \to \infty$, the rate of convergence comes within a logarithmic factor of the parametric rate $n^{-1/2}$. The proof is discussed in the final section.

## 5 Efficient Algorithms

The optimally pruned subtree $T_n$ of rule (2) can be computed exactly in $O(|T_0|^2)$ operations. This follows from a simple bottom-up dynamic programming algorithm, which we

describe below, and uses a method for "square-root" pruning studied in [7]. In the context of Theorem 2, we have $|T_0| = m^d \sim n$, so the algorithm runs in time $O(n^2)$.

Note that an algorithm for finding the optimal $R \leq T$ was provided in [4]. We now describe an algorithm for finding both the optimal $T \leq T_0$ and $R \leq T$ solving (2). Given a node $v \in T_0$, let $T_v^*$ be the subtree of $T_0$ rooted at $v$ that minimizes the objective function of (2), and let $R_v^*$ be the associated subtree that minimizes $\Delta(T_v^*, S, R)$. The problem is solved by finding $T_{\text{root}}^*$ and $R_{\text{root}}^*$ using a bottom-up procedure.

If $v$ is a leaf node of $T_0$, then clearly $T_v^* = R_v^* = \{v\}$. If $v$ is an internal node, denote the children of $v$ by $u$ and $w$. There are three cases for $T_v^*$ and $R_v^*$: (i) $|T_v^*| = |R_v^*| = 1$, in which case $T_v^* = R_v^* = \{v\}$; (ii) $|T_v^*| \geq |R_v^*| > 1$, in which case $T_v^*$ and $R_v^*$ can be computed by merging $T_u^*$ with $T_w^*$ and $R_u^*$ with $R_w^*$, respectively; (iii) $|T_v^*| > |R_v^*| = 1$, in which case $R_v^* = \{v\}$, and $T_v^*$ is determined by solving a square root pruning problem, just like the one in (1). At each node, these three candidates are determined, and $T_v^*$ and $R_v^*$ are the candidates minimizing the objective function (empirical error plus penalty) at each node. Using the first algorithm in [7], the overall pruning procedure may be accomplished in $(|T_0|^2)$ operations.

Determining the optimally pruned degree $r$ PDCT is more challenging. The problem requires the construction, at each node of $T_0$, a polynomial classifier of degree $r$ having minimum empirical error. Unfortunately, this task is computationally infeasible for large sample sizes. As an alternative, we recommend the use of polynomial support vector machines. SVMs are well known for their good generalization ability in practical problems. Moreover, linear SVMs in perceptron trees have been shown to work well [6].

## 6  Conclusions

A key aspect of DCTs is their spatial adaptivity, which enables local (rather than global) fitting of the decision boundary. Our risk analysis involves a spatial decomposition of the usual concentration inequalities, leading to a spatially adaptive, data-dependent pruning criterion that promotes unbalanced trees that focus on the decision boundary. For distributions on $(X, Y)$ whose Bayes decision boundary behave locally like a Hölder-$\gamma$ smooth function, we show that the PDCT error converges to the Bayes error at a rate no slower than $O((\log n/n)^{\gamma/(d+2\gamma-2)})$. Polynomial kernel methods are not capable of achieving such rates due to their lack of spatial adaptivity. When $\gamma = 1$, the DCT convergence rate is within a logarithmic factor of the minimax optimal rate. As $\gamma \to \infty$ the rate tends to within a log factor of $n^{-1/2}$, the parametric rate. However, the rates for $\gamma > 1$ are not within a logarithmic factor of the minimax rate [2]. It may be possible to tighten the bounds further. On the other hand, near-minimax rates might not be achievable using rectangular partitions, and more flexible partitioning schemes, such as adaptive triangulations, may be required.

## 7  Proof Sketches

The key to proving Theorem 1 is the following result, which is a modified version of a theorem of Mansour and McAllester [4].

**Lemma 1** *Let $\delta \in (0, 1)$. With probability at least $1 - \delta$, every $T \leq T_0$ satisfies*

$$\epsilon(T) \leq \hat{\epsilon}(T) + \min_{R \leq T} f(T, S, R, \delta),$$

*where*

$$f(T, S, R, \delta) = \sum_{v \in L(R)} \frac{1}{n} \left[ \sqrt{48 n_v |T_v| \log(2n)} \right.$$

$$+ \sqrt{24 n_v [d \log(m) + \log(3/\delta)]} + 2[d \log(m) + \log(3/\delta)] \Big]$$

Our primary modification to the lemma is to replace one local uniform deviation inequality (which holds for countable collections of classifiers [4, Lemma 4]) with another (which holds for infinite collections of classifiers [3, Lemma 2]). This eases our extension to polynomial-decorated DCTs in Section 4, by allowing us to avoid tedious quantization arguments.

To prove Theorem 1, define the event $\Omega_m$ to be the collection of all training samples $S$ such that for all $T \leq T_0$, the bound of Lemma 1 holds, with $\delta = 3/m^d$. By that lemma, $\mathbf{P}(\Omega_m) \geq 1 - 3/m^d$. Let $T \leq T_0$ be arbitrary. We have

$$
\begin{aligned}
&\mathbf{E}\{\epsilon(T_n) - \epsilon(T)\} \\
&= \quad P(\Omega_m)\mathbf{E}\{\epsilon(T_n) - \epsilon(T) \,|\, \Omega_m\} + P(\Omega_m^c)\mathbf{E}\{\epsilon(T_n) - \epsilon(T) \,|\, \Omega_m^c\} \\
&\leq \quad \mathbf{E}\{\epsilon(T_n) - \epsilon(T) \,|\, \Omega_m\} + \frac{3}{m^d}.
\end{aligned}
$$

Given $S \in \Omega_m$, we know

$$
\begin{aligned}
\epsilon(T_n) &\leq \quad \hat{\epsilon}(T_n) + \min_{R \leq T_n} f(T_n, S, R, 3m^{-d}) \\
&= \quad \hat{\epsilon}(T_n) + \min_{R \leq T_n} \Delta(T_n, S, R) + \frac{4 d \log(m)}{n} \\
&\leq \quad \hat{\epsilon}(T) + \min_{R \leq T} \Delta(T, S, R) + \frac{4 d \log(m)}{n},
\end{aligned}
$$

where the last inequality comes from the definition of $T_n$.

From Chernoff's inequality, we know $\mathbf{P}\{\hat{\epsilon}(T) \geq \epsilon(T) + t\} \leq e^{-2nt^2}$. By applying this bound, and the fact $\mathbf{E}\{Z\} \leq \int_0^\infty \mathbf{P}\{Z > t\}\, dt$, the theorem is proved. $\qquad \square$

## 7.1 Proof of Theorem 2

By Theorem 1, it suffices to find a tree $T^* \leq T_0$ such that

$$\mathbf{E}\left[\min_{R \leq T^*} \Delta(T^*, S, R)\right] + (\epsilon(T^*) - \epsilon^*) = O\left(\left(\frac{\log n}{n}\right)^{\frac{1}{d}}\right).$$

Define $T^*$ to be the tree obtained by pruning back $T_0$ at every node (thought of as a region of space) that does not intersect the Bayes decision boundary. It can be shown without much difficulty that $\epsilon(T^*) - \epsilon^* = O((\log n/n)^{1/d})$ [9, Lemma 1]. It remains to bound the estimation error.

Recall that $T_0$ (and hence $T^*$) has depth $Jd$, where $J = \log_2(m)$. Define $R^*$ to be the pruned subtree of $T^*$ consisting of all nodes in $T^*$ up to depth $j_0 d$, where $j_0 = J - (1/d)\log_2(J)$ (truncated if necessary). Let $\Omega_v$ be the set of all training samples such that $\sqrt{n_v} \leq 2\sqrt{np_v}$. Let $\Omega$ be the set of all training samples $S$ such that $S \in \Omega_v$ for all $v \in L(R^*)$.

Now

$$
\begin{aligned}
&\mathbf{E}\left[\min_{R \leq T^*} \Delta(T^*, S, R)\right] \\
&\leq \quad \mathbf{P}(\Omega)\mathbf{E}\left[\min_{R \leq T^*} \Delta(T^*, S, R)|\Omega\right] + \mathbf{P}(\Omega^c)\mathbf{E}\left[\min_{R \leq T^*} \Delta(T^*, S, R)|\Omega^c\right].
\end{aligned}
$$

It can be shown, by applying the union bound, A2, and a theorem of Okamoto [10], that $\mathbf{P}(\Omega^c) = O((\log n/n)^{1/d})$. Moreover, the second expectation on the right is easily seen to be $O(1)$ by considering the root fragment consisting of only the root node. Hence it remains to bound the first term on the right-hand side. We use $\mathbf{P}(\Omega) \leq 1$, and focus on bounding the expectation. It can be shown, assuming $S \in \Omega$, that $\Delta(T^*, S, R^*) = O((\log n/n)^{1/d})$. It suffices to bound the first term of $\Delta(T^*, S, R^*)$, which clearly dominates the second term. The first term, consisting of a sum of terms over the leaf nodes of $R^*$, is dominated by the sum of those terms over the leaf nodes of $R^*$ at depth $j_0 d$. The number of such nodes may be bounded by assumption A2. The remaining expression is bounded using assumptions A1 and A2, as well as the definitions of $T^*$, $R^*$, and $\Omega$.

## 7.2 Proof of Theorem 3

The estimation error is increased by a constant $\propto \sqrt{V_{d,r}}$, so its asymptotic analysis remains unchanged. The only significant change is in the analysis of the approximation error. The tree $T^*$ is defined as in the previous proof. Recall the leaf nodes of $T^*$ at maximum depth are cells of side length $1/m$. By a simple Taylor series argument, the approximation error $\epsilon(T^*) - \epsilon^*$ behaves like $m^{-\gamma}$. The remainder of the proof is essentially the same as the proof of Theorem 2.

## Acknowledgments

This work was partially supported by the National Science Foundation, grant nos. MIP–9701692 and ANI-0099148, the Army Research Office, grant no. DAAD19-99-1-0349, and the Office of Naval Research, grant no. N00014-00-1-0390.

## References

[1] C. Scott and R. Nowak, "Dyadic classification trees via structural risk minimization," in *Advances in Neural Information Processing Systems 14*, S. Becker, S. Thrun, and K. Obermayer, Eds., Cambridge, MA, 2002, MIT Press.

[2] E. Mammen and A. B. Tsybakov, "Smooth discrimination analysis," *Annals of Statistics*, vol. 27, pp. 1808–1829, 1999.

[3] M. Kearns and Y. Mansour, "A fast, bottom-up decision tree pruning algorithm with near-optimal generalization," in *International Conference on Machine Learning*, 1998, pp. 269–277.

[4] Y. Mansour and D. McAllester, "Generalization bounds for decision trees," in *Proceedings of the Thirteenth Annual Conference on Computational Learning Theory, Palo Alto, California*, Nicolò Cesa-Bianchi and Sally A. Goldman, Eds., 2000, pp. 69–74.

[5] K. Bennett and J. Blue, "A support vector machine approach to decision trees," in *Proceedings of the IEEE International Joint Conference on Neural Networks, Anchorage, Alaska*, 1998, vol. 41, pp. 2396–2401.

[6] K. Bennett, N. Cristianini, J. Shawe-Taylor, and D. Wu, "Enlarging the margins in perceptron decision trees," *Machine Learning*, vol. 41, pp. 295–313, 2000.

[7] C. Scott, "Tree pruning using a non-additive penalty," Tech. Rep. TREE 0301, Rice University, 2003, available at http://www.dsp.rice.edu/~cscott/pubs.html.

[8] A. Barron, "Complexity regularization with application to artificial neural networks," in *Nonparametric functional estimation and related topics*, G. Roussas, Ed., pp. 561–576. NATO ASI series, Kluwer Academic Publishers, Dordrecht, 1991.

[9] C. Scott and R. Nowak, "Complexity-regularized dyadic classification trees: Efficient pruning and rates of convergence," Tech. Rep. TREE0201, Rice University, 2002, available at http://www.dsp.rice.edu/~cscott/pubs.html.

[10] M. Okamoto, "Some inequalities relating to the partial sum of binomial probabilites," *Annals of the Institute of Statistical Mathematics*, vol. 10, pp. 29–35, 1958.
